# Learning a Hierarchical Belief Network of Independent Factor Analyzers

**H. Attias**[*]
hagai@gatsby.ucl.ac.uk
Sloan Center for Theoretical Neurobiology, Box 0444
University of California at San Francisco
San Francisco, CA 94143-0444

## Abstract

Many belief networks have been proposed that are composed of binary units. However, for tasks such as object and speech recognition which produce real-valued data, binary network models are usually inadequate. Independent component analysis (ICA) learns a model from real data, but the descriptive power of this model is severly limited. We begin by describing the independent factor analysis (IFA) technique, which overcomes some of the limitations of ICA. We then create a multilayer network by cascading single-layer IFA models. At each level, the IFA network extracts real-valued latent variables that are non-linear functions of the input data with a highly adaptive functional form, resulting in a hierarchical distributed representation of these data. Whereas exact maximum-likelihood learning of the network is intractable, we derive an algorithm that maximizes a lower bound on the likelihood, based on a variational approach.

## 1  Introduction

An intriguing hypothesis for how the brain represents incoming sensory information holds that it constructs a hierarchical probabilistic model of the observed data. The model parameters are learned in an unsupervised manner by maximizing the likelihood that these data are generated by the model. A multilayer belief network is a realization of such a model. Many belief networks have been proposed that are composed of binary units. The hidden units in such networks represent latent variables that explain different features of the data, and whose relation to the

---

[*]Current address: Gatsby Computational Neuroscience Unit, University College London, 17 Queen Square, London WC1N 3AR, U.K.

data is highly non-linear. However, for tasks such as object and speech recognition which produce real-valued data, the models provided by binary networks are often inadequate. Independent component analysis (ICA) learns a generative model from real data, and extracts real-valued latent variables that are mutually statistically independent. Unfortunately, this model is restricted to a single layer and the latent variables are simple linear functions of the data; hence, underlying degrees of freedom that are non-linear cannot be extracted by ICA. In addition, the requirement of equal numbers of hidden and observed variables and the assumption of noiseless data render the ICA model inappropriate.

This paper begins by introducing the independent factor analysis (IFA) technique. IFA is an extension of ICA, that allows different numbers of latent and observed variables and can handle noisy data. The paper proceeds to create a multilayer network by cascading single-layer IFA models. The resulting generative model produces a hierarchical distributed representation of the input data, where the latent variables extracted at each level are *non-linear* functions of the data with a highly adaptive functional form. Whereas exact maximum-likelihood (ML) learning in this network is intractable due to the difficulty in computing the posterior density over the hidden layers, we present an algorithm that maximizes a lower bound on the likelihood. This algorithm is based on a general variational approach that we develop for the IFA network.

## 2 Independent Component and Independent Factor Analysis

Although the concept of ICA originated in the field of signal processing, it is actually a density estimation problem. Given an $L' \times 1$ observed data vector $\mathbf{y}$, the task is to explain it in terms of an $L \times 1$ vector $\mathbf{x}$ of unobserved 'sources' that are mutually statistically independent. The relation between the two is assumed linear,

$$\mathbf{y} = \mathbf{Hx} + \mathbf{u} , \tag{1}$$

where $\mathbf{H}$ is the 'mixing' matrix; the noise vector $\mathbf{u}$ is usually assumed zero-mean Gaussian with a covariance matrix $\mathbf{\Lambda}$. In the context of blind source separation [1]-[4], the source signals $\mathbf{x}$ should be recovered from the mixed noisy signals $\mathbf{y}$ with no knowledge of $\mathbf{H}$, $\mathbf{\Lambda}$, or the source densities $p(x_i)$, hence the term 'blind'. In the density estimation approach, one regards (1) as a probabilistic generative model for the observed $p(\mathbf{y})$, with the mixing matrix, noise covariance, and source densities serving as model parameters. In principle, these parameters should be learned by ML, followed by inferring the sources via a MAP estimator.

For Gaussian sources, (1) is the factor analysis model, for which an EM algorithm exists and the MAP estimator is linear. The problem becomes interesting and more difficult for non-Gaussian sources. Most ICA algorithms focus on square ($L' = L$), noiseless ($\mathbf{y} = \mathbf{Hx}$) mixing, and fix $p(x_i)$ using prior knowledge (but see [5] for the case of noisy mixing with a fixed Laplacian source prior). Learning $\mathbf{H}$ occurs via gradient-ascent maximization of the likelihood [1]-[4]. Source density parameters can also be adapted in this way [3],[4], but the resulting gradient-ascent learning is rather slow. This state of affairs presented a problem to ICA algorithms, since the ability to learn arbitrary source densities that are not known in advance is crucial: using an inaccurate $p(x_i)$ often leads to a bad $\mathbf{H}$ estimate and failed separation.

This problem was recently solved by introducing the IFA technique [6]. IFA employs a semi-parametric model of the source densities, which allows learning them (as well as the mixing matrix) using expectation-maximization (EM). Specifically, $p(x_i)$ is described as a mixture of Gaussians (MOG), where the mixture

components are labeled by $s = 1, ..., n_i$ and have means $\mu_{i,s}$ and variances $\gamma_{i,s}$: $p(x_i) = \sum_s p(s_i = s) \mathcal{G}(x_i - \mu_{i,s}, \gamma_{i,s})$. [1] The mixing proportions are parametrized using the softmax form: $p(s_i = s) = \exp(a_{i,s}) / \sum_{s'} \exp(a_{i,s'})$. Beyond noiseless ICA, an EM algorithm for the noisy case (1) with any $L, L'$ was also derived in [6] using the MOG description. [2] This algorithm learns a probabilistic model $p(\mathbf{y} \mid W)$ for the observed data, parametrized by $W = (\mathbf{H}, \mathbf{\Lambda}, \{a_{i,s}, \mu_{i,s}, \gamma_{i,s}\})$. A graphical representation of this model is provided by Fig. 1, if we set $n = 1$ and $y_j^0 = b_{j,s}^1 = \nu_{j,s}^1 = 0$.

## 3 Hierarchical Independent Factor Analysis

In the following we develop a multilayer generalization of IFA, by cascading duplicates of the generative model introduced in [6]. Each layer $n = 1, ..., N$ is composed of two sublayers: a source sublayer which consists of the units $x_i^n$, $i = 1, ..., L_n$, and an output sublayer which consists of $y_j^n$, $j = 1, ..., L_n'$. The two are linearly related via $\mathbf{y}^n = \mathbf{H}^n \mathbf{x}^n + \mathbf{u}^n$ as in (1); $\mathbf{u}^n$ is a Gaussian noise vector with covariance $\mathbf{\Lambda}^n$. The $n$th-layer source $x_i^n$ is described by a MOG density model with parameters $a_{i,s}^n$, $\mu_{i,s}^n$, and $\gamma_{i,s}^n$, in analogy to the IFA sources above.

The important step is to determine how layer $n$ depends on the previous layers. We choose to introduce a dependence of the $i$th source of layer $n$ only on the $i$th output of layer $n - 1$. Notice that matching $L_n = L_{n-1}'$ is now required. This dependence is implemented by making the means and mixture proportions of the Gaussians which compose $p(x_i^n)$ dependent on $y_i^{n-1}$. Specifically, we make the replacements $\mu_{i,s}^n \to \mu_{i,s}^n + \nu_{i,s}^n y_i^{n-1}$ and $a_{i,s}^n \to a_{i,s}^n + b_{i,s}^n y_i^{n-1}$. The resulting joint density for layer $n$, conditioned on layer $n - 1$, is

$$p(\mathbf{s}^n, \mathbf{x}^n, \mathbf{y}^n \mid \mathbf{y}^{n-1}, W^n) = \prod_{i=1}^{L_n} p(s_i^n \mid y_i^{n-1}) \, p(x_i^n \mid s_i^n, y_i^{n-1}) \, p(\mathbf{y}^n \mid \mathbf{x}^n) \,, \qquad (2)$$

where $W^n$ are the parameters of layer $n$ and

$$p(s_i^n = s \mid y_i^{n-1}) = \frac{\exp(a_{i,s}^n + b_{i,s}^n y_i^{n-1})}{\sum_{s'} \exp(a_{i,s'}^n + b_{i,s'}^n y_i^{n-1})} \,, \qquad p(\mathbf{y}^n \mid \mathbf{x}^n) = \mathcal{G}(\mathbf{y}^n - \mathbf{H}^n \mathbf{x}^n, \mathbf{\Lambda}^n) \,,$$

$$p(x_i^n \mid s_i^n = s, y_i^{n-1}) = \mathcal{G}(x_i^n - \mu_{i,s}^n - \nu_{i,s}^n y_i^{n-1}, \gamma_{i,s}^n) \,.$$

The full model joint density is given by the product of (2) over $n = 1, ..., N$ (setting $\mathbf{y}^0 = 0$). A graphical representation of layer $n$ of the hierarchical IFA network is given in Fig. 1. All units are hidden except $\mathbf{y}^N$.

To gain some insight into our network, we examine the relation between the $n$th-layer source $x_i^n$ and the $n - 1$th-layer output $y_i^{n-1}$. This relation is probabilistic and is determined by the conditional density $p(x_i^n \mid y_i^{n-1}) = \sum_{s_i^n} p(s_i^n \mid y_i^{n-1}) p(x_i^n \mid s_i^n, y_i^{n-1})$. Notice from (2) that this is a MOG density. Its $y_i^{n-1}$-dependent mean is given by

$$\overline{x_i^n} = f_i^n(y_i^{n-1}) = \sum_s p(s_i^n = s \mid y_i^{n-1}) \, (\mu_{i,s}^n + \nu_{i,s}^n y_i^{n-1}) \,, \qquad (3)$$

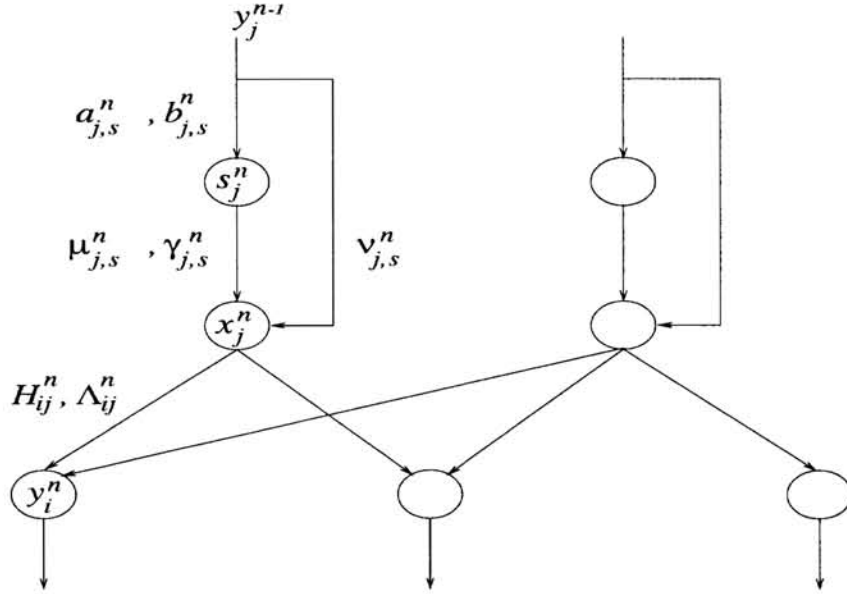

Figure 1: Layer $n$ of the hierarchical ICA generative model.

and is a non-linear function of $y_i^{n-1}$ due to the softmax form of $p(s_i^n \mid y_i^{n-1})$. By adjusting the parameters, the function $f_i^n$ can assume a very wide range of forms: suppose that for state $s_i^n$, $a_{i,s}^n$ and $b_{i,s}^n$ are set so that $p(s_i^n = s \mid y_i^{n-1})$ is significant only in a small, continuous range of $y_i^{n-1}$ values, with different ranges associated with different $s$'s. In this range, $f_i^n$ will be dominated by the linear term $\mu_{i,s}^n + \nu_{i,s}^n y_i^{n-1}$. Hence, a desired $f_i^n$ can be produced by placing oriented line segments at appropriate points above the $y_i^{n-1}$-axis, then smoothly join them together by the $p(s_i^n \mid y_i^{n-1})$. Using the algorithm below, the optimal form of $f_i^n$ will be learned from the data. Therefore, our model describes the data $y_i^N$ as a potentially highly complex function of the top layer sources, produced by repeated application of linear mixing followed by a non-linearity, with noise allowed at each stage.

## 4   Learning and Inference by Variational EM

The need for summing over an exponentially large number of source state configurations $(s_1^n, ..., s_L^n)$, and integrating over the softmax functions $p(s_i^n \mid y_i^n)$, makes exact learning intractable in our network. Thus, approximations must be made. In the following we develop a variational approach, in the spirit of [8], to hierarchical IFA. We begin, following the approach of [7] to EM, by bounding the log-likelihood from below: $\mathcal{L} = \log p(\mathbf{y}^N) \geq \sum_n \{E \log p(\mathbf{y}^n \mid \mathbf{x}^n) + \sum_{i,s_i^n} [E \log p(x_i^n \mid s_i^n, y_i^{n-1}) + E \log p(s_i^n \mid y_i^{n-1})]\} - E \log q$, where $E$ denotes averaging over the hidden layers using an arbitrary posterior $q = q(\mathbf{s}^{1 \cdots N}, \mathbf{x}^{1 \cdots N}, \mathbf{y}^{1 \cdots N-1} \mid \mathbf{y}^N)$. In exact EM, $q$ at each iteration is the true posterior, parametrized by $W^{1 \cdots N}$ from the previous iteration. In variational EM, $q$ is chosen to have a form which makes learning tractable, and is parametrized by a separate set of parameters $V^{1 \cdots N}$. These are optimized to bring $q$ as close to the true posterior as possible.

**E-step.** We use a variational posterior that is factorized across layers. Within layer $n$ it has the form

$$q(\mathbf{s}^n, \mathbf{x}^n, \mathbf{y}^n \mid V^n) = \prod_{i=1}^{L_n} v_{i,s_i}^n \; \mathcal{G}(\mathbf{z}^n - \boldsymbol{\rho}^n, \boldsymbol{\Sigma}^n) \,, \qquad \mathbf{z}^n = (\mathbf{x}^n, \mathbf{y}^n)^T \qquad (4)$$

for $n < N$, and $q(\mathbf{s}^N, \mathbf{x}^N \mid V^N) = \prod_i v_{i,s_i}^N \mathcal{G}(\mathbf{x}^N - \boldsymbol{\rho}^N, \boldsymbol{\Sigma}^N)$. The variational parameters $V^n = (\boldsymbol{\rho}^n, \boldsymbol{\Sigma}^n, \{v_{i,s}^n\})$ depend on the data $\mathbf{y}^N$. The full $N$-layer posterior is simply a product of (4) over $n$. Hence, given the data, the $n$th-layer sources and outputs are jointly Gaussian whereas the states $s_i^n$ are independent. [3]

Even with the variational posterior (4), the term $E \log p(s_i^n \mid y_i^{n-1})$ in the lower bound cannot be calculated analytically, since it involves integration over the softmax function. Instead, we calculate yet a lower bound on this term. Let $c_{i,s}^n = a_{i,s}^n + b_{i,s}^n y_i^{n-1}$ and drop the unit and layer indices $i, n$, then $\log p(s \mid y) = -\log(1 + e^{-c_s} \sum_{s' \neq s} e^{c_{s'}})$. Borrowing an idea from [8], we multiply and divide by $e^{\eta_s}$ under the logarithm sign and use Jensen's inequality to get $E \log p(s \mid y) \geq -\eta_s E c_s - \log E \left[ e^{-\eta_s c_s} + e^{-(1+\eta_s)c_s} \sum_{s' \neq s} e^{c_{s'}} \right]$. This results in a bound that can be calculated in closed form:

$$E \log p(s_i^n = s \mid y_i^{n-1}) \geq -v_s^n \eta_s^n \bar{c}_s^n - v_s^n \log \left( e^{f_s^n} + \sum_{s' \neq s} e^{f_{ss'}^n} \right) \equiv \mathcal{F}_{i,s}^n \,, \qquad (5)$$

where $\bar{c}_s^n = a_s^n + b_s^n \rho_y^{n-1}$, $f_s^n = -\eta_s^n \bar{c}_s^n + (\eta_s^n b_s^n)^2 \Sigma_{yy}^{n-1}/2$, $f_{ss'}^n = -(1+\eta_s^n)\bar{c}_s^n + \bar{c}_{s'}^n + [(1+\eta_s^n)b_s^n - b_{s'}^n]^2 \Sigma_{yy}^{n-1}/2$, and the subscript $i$ is omitted. We also defined $\boldsymbol{\rho}^n = (\boldsymbol{\rho}_x^n, \boldsymbol{\rho}_y^n)^T$ and similarly $\boldsymbol{\Sigma}_{xx}, \boldsymbol{\Sigma}_{yy}, \boldsymbol{\Sigma}_{xy} = \boldsymbol{\Sigma}_{yx}^T$ are the subblocks of $\boldsymbol{\Sigma}$. Since (5) holds for arbitrary $\eta_{i,s}^n$, the latter are treated as additional variational parameters which are optimized to tighten this bound. [4]

To optimize the variational parameters $V^{1 \cdots N}$, we equate the gradient of the lower bound on $\mathcal{L}$ to zero and obtain

$$\begin{pmatrix} (\mathbf{H}^T \boldsymbol{\Lambda}^{-1} \mathbf{H})^n + \mathbf{A}^n & -(\mathbf{H}^T \boldsymbol{\Lambda}^{-1})^n \\ -(\boldsymbol{\Lambda}^{-1} \mathbf{H})^n & (\boldsymbol{\Lambda}^{-1})^n + \mathbf{B}^{n+1} \end{pmatrix} \boldsymbol{\rho}^n - \begin{pmatrix} 0 & \mathbf{B}^n \\ \mathbf{A}^{n+1} & 0 \end{pmatrix} \begin{pmatrix} \rho_x^{n+1} \\ \rho_y^{n-1} \end{pmatrix}$$

$$= \begin{pmatrix} \alpha^n \\ \beta^{n+1} + \mathcal{F}_\rho^{n+1} \end{pmatrix} \,, \qquad (6)$$

$$\boldsymbol{\Sigma}^n = \begin{pmatrix} (\mathbf{H}^T \boldsymbol{\Lambda}^{-1} \mathbf{H})^n + \mathbf{A}^n & -(\mathbf{H}^T \boldsymbol{\Lambda}^{-1})^n \\ -(\boldsymbol{\Lambda}^{-1} \mathbf{H})^n & (\boldsymbol{\Lambda}^{-1})^n + \mathbf{B}^{n+1} - \mathcal{F}_\Sigma^{n+1} \end{pmatrix}^{-1} \,, \qquad (7)$$

where $A_{ij}^n = \sum_s (v_{i,s}/\gamma_{i,s})^n \delta_{ij}$, $B_{ij}^n = \sum_s (v_{i,s}\nu_{i,s}/\gamma_{i,s})^n \delta_{ij}$, $\alpha_i^n = \sum_s (v_{i,s}\mu_{i,s}/\gamma_{i,s})^n$, and $\beta_i^n = \sum_s (v_{i,s}\mu_{i,s}\nu_{i,s}/\gamma_{i,s})^n$. (All parameters within $(\cdots)^n$ belong to layer $n$). $F_{\rho,\Sigma}^{n+1}$ contain the corresponding derivatives of $\mathcal{F}_s^{n+1}$ (5), summed over $s$. For the state posteriors we have

$$v_s^n = \frac{1}{Z^n} \exp \left( \frac{\gamma_s^n}{2} + \frac{1}{2\gamma_s^n}[(\rho_x^n - \mu_s^n - \nu_s^n \rho_y^{n-1})^2 + \Sigma_{xx}^n + (\nu_s^n)^2 \Sigma_{yy}^{n-1}] + \frac{\partial \mathcal{F}_s^n}{\partial v_s^n} \right) \,, \qquad (8)$$

where the unit subscript $i$ is omitted (i.e., $\Sigma_{xx}^n = \Sigma_{xx,ii}^n$); $Z^n = Z_i^n$ is set such that $\sum_s v_{i,s}^n = 1$. A simple modification of these equations is required for layer $n = N$.

The optimal $V^{1\cdots N}$ are obtained by solving the fixed-point equations (6–8) iteratively for each data vector $\mathbf{y}^N$, keeping the generative parameters $W^{1\cdots N}$ fixed. Notice that these equations couple layer $n$ to layers $n \pm 1$. The additional parameters $\eta_{i,s}^n$ are adjusted using gradient ascent on $\mathcal{F}_{i,s}^n$. Once learning is complete, the inference problem is solved since the MAP estimate of the hidden unit values given the data is readily available from $\rho_i^n$ and $v_{i,s}^n$.

**M-Step.** In terms of the variational parameters obtained in the E-step, the new generative parameters are given by

$$
\begin{aligned}
\mathbf{H}^n &= (\rho_y^n \rho_x^{n\,T} + \Sigma_{yx}^n)(\rho_x^n \rho_x^{n\,T} + \Sigma_{xx}^n)^{-1}\,,\\
\Lambda^n &= \rho_y^n \rho_y^{n\,T} + \Sigma_{yy}^n - \mathbf{H}^n(\rho_x^n \rho_x^{n\,T} + \Sigma_{xy}^n)\,,
\end{aligned}
\tag{9}
$$

$$
\begin{aligned}
\begin{pmatrix} \mu_s^n \\ \nu_s^n \end{pmatrix} &= \begin{pmatrix} v_s^n & \rho_y^{n-1} v_s^n \\ \rho_y^{n-1} v_s^n & [(\rho_y^{n-1})^2 + \Sigma_{yy}^{n-1}]v_s^n \end{pmatrix}^{-1} \begin{pmatrix} \rho_x^n v_s^n \\ \rho_x^n \rho_y^{n-1} v_s^n \end{pmatrix}\,,\\
\gamma_s^n &= \frac{1}{v_s^n}\left[(\rho_x^n - \mu_s^n - \nu_s^n \rho_y^{n-1})^2 + \Sigma_{xx}^n + (\nu_s^n)^2 \Sigma_{yy}^{n-1}\right]v_s^n\,,
\end{aligned}
\tag{10}
$$

omitting the subscript $i$ as in (8), and are slightly modified for layer $N$. In batch mode, averaging over the data is implied and the $v_s^n$ do not cancel out. Finally, the softmax parameters $a_{i,s}^n, b_{i,s}^n$ are adapted by gradient ascent on the bound (5).

## 5   Discussion

The hierarchical IFA network presented here constitutes a quite general framework for learning and inference using real-valued probabilistic models that are strongly non-linear but highly adaptive. Notice that this network includes both continuous $x_i^n, y_i^n$ and binary $s_i^n$ units, and can thus extract both types of latent variables. In particular, the uppermost units $s_i^1$ may represent class labels in classification tasks. The models proposed in [9]-[11] can be viewed as special cases where $x_i^n$ is a prescribed deterministic function (e.g., rectifier) of the previous outputs $y_j^{n-1}$: in the IFA network, a deterministic (but still adaptive) dependence can be obtained by setting the variances $\gamma_{i,s}^n = 0$. Note that the source $x_i^n$ in such a case assumes only the values $\mu_{i,s}^n$, and thus corresponds to a discrete latent variable.

The learning and inference algorithm presented here is based on the variational approach. Unlike variational approximations in other belief networks [8],[10] which use a completely factorized approximation, the structure of the hierarchical IFA network facilitates using a variational posterior that allows correlations among hidden units occupying the same layer, thus providing a more accurate description of the true posterior. It would be interesting to compare the performance of our variational algorithm with the belief propagation algorithm [12] which, when adapted to the densely connected IFA network, would also be an approximation. Markov chain Monte Carlo methods, including the more recent slice sampling procedure used in [11], would become very slow as the network size increases.

It is possible to consider a more general non-linear network along the lines of hierarchical IFA. Notice from (2) that given the previous layer output $\mathbf{y}^{n-1}$, the mean output of the next layer is $\overline{y_i^n} = \sum_j H_{ij}^n f_j^n(y_j^{n-1})$ (see (3)), i.e. a linear mixing preceded by a non-linear function operating on each output component separately. However, if we eliminate the sources $x_j^n$, replace the individual source

states $s_j^n$ by collective states $s^n$, and allow the linear transformation to depend on $s^n$, we arrive at the following model: $p(s^n = s \mid \mathbf{y}^{n-1}) \propto \exp(\underline{a_s^n} + \mathbf{b}_s^{n}{}^T \mathbf{y}^{n-1})$, $p(\mathbf{y}^n \mid s^n = s, \mathbf{y}^{n-1}) = \mathcal{G}(\mathbf{y}^n - \mathbf{h}_s^n - \mathbf{H}_s^n \mathbf{y}^{n-1}, \mathbf{\Lambda}^n)$. Now we have $\overline{\mathbf{y}^n} = \sum_s p(s^n = s \mid \mathbf{y}^{n-1})(\mathbf{h}_s^n + \mathbf{H}_s^n \mathbf{y}^{n-1}) \equiv F(\mathbf{y}^{n-1})$, which is a more general non-linearity.

Finally, the blocks $\{\mathbf{y}^n, \mathbf{x}^n, s^n \mid \mathbf{y}^{n-1}\}$ (Fig. 1), or alternatively the blocks $\{\mathbf{y}^n, s^n \mid \mathbf{y}^{n-1}\}$ described above, can be connected not only vertically (as in this paper) and horizontally (creating layers with multiple blocks), but in any directed acyclic graph architecture, with the variational EM algorithm extended accordingly.

## Acknowledgements

I thank V. de Sa for helpful discussions. Supported by The Office of Naval Research (N00014-94-1-0547), NIDCD (R01-02260), and the Sloan Foundation.

## Footnotes

[1]Throughout this paper, $\mathcal{G}(\mathbf{x}, \mathbf{\Sigma}) = \mid 2\pi \mathbf{\Sigma} \mid^{-1/2} \exp(-\mathbf{x}^T \mathbf{\Sigma}^{-1} \mathbf{x}/2)$.

[2]However, for many sources the E-step becomes intractable, since the number $\prod_i n_i$ of source state configurations $\mathbf{s} = (s_1, ..., s_L)$ depends exponentially on $L$. Such cases are treated in [6] using a variational approximation.

[3] It is easy to introduce more structure into (4) by allowing the means $\rho_i^n$ to depend on $s_i^n$, and the covariances $\Sigma_{ij}^n$ to depend on $s_i^n, s_j^n$, thus making the approximation more accurate (but more complex) while maintaining tractability.

[4] An alternative approach to handle $E \log p(s_i^n \mid y_i^{n-1})$ is to approximate the required integral by, e.g., the maximum value of the integrand, possibly including Gaussian corrections. The resulting approximation is simpler than (5); however, it is no longer guaranteed to bound the log-likelihood from below.

## References

[1] Bell, A.J. and Sejnowski, T.J. (1995). An information-maximization approach to blind separation and blind deconvolution. *Neural Computation* **7**, 1129-1159.

[2] Cardoso, J.-F. (1997). Infomax and maximum likelihood for source separation. *IEEE Signal Processing Letters* **4**, 112-114.

[3] Pearlmutter, B.A. and Parra, L.C. (1997). Maximum likelihood blind source separation: A context-sensitive generalization of ICA. *Advances in Neural Information Processing Systems* **9** (Ed. Mozer, M.C. et al), 613-619. MIT Press.

[4] Attias, H. and Schreiner, C.E. (1998). Blind source separation and deconvolution: the dynamic component analysis algorithm. *Neural Computation* **10**, 1373-1424.

[5] Lewicki, M.S. and Sejnowski, T.J. (1998). Learning nonlinear overcomplete representations for efficient coding. *Advances in Neural Information Processing Systems* **10** (Ed. Jordan, M.I. et al), MIT Press.

[6] Attias, H. (1999). Independent factor analysis. *Neural Computation*, in press.

[7] Neal, R.M. and Hinton, G.E. (1998). A view of the EM algorithm that justifies incremental, sparse, and other variants. *Learning in Graphical Models* (Ed. Jordan, M.I.), Kluwer Academic Press.

[8] Saul, L.K., Jaakkola, T., and Jordan, M.I. (1996). Mean field theory of sigmoid belief networks. *Journal of Artificial Intelligence Research* **4**, 61-76.

[9] Frey, B.J. (1997) Continuous sigmoidal belief networks trained using slice sampling. *Advances in Neural Information Processing Systems* **9** (Ed. Mozer, M.C. et al). MIT Press.

[10] Frey, B.J. and Hinton, G.E. (1999). Variational learning in non-linear Gaussian belief networks. *Neural Computation*, in press.

[11] Ghahramani, Z. and Hinton, G.E. (1998). Hierarchical non-linear factor analysis and topographic maps. *Advances in Neural Information Processing Systems* **10** (Ed. Jordan, M.I. et al), MIT Press.

[12] Pearl, J. (1988). *Probabilistic Reasoning in Intelligent Systems*. Morgan Kaufmann, San Mateo, CA.